# Unsupervised Feature Selection for Accurate Recommendation of High-Dimensional Image Data

**Sabri Boutemedjet**
DI, Universite de Sherbrooke
2500 boulevard de l'Université
Sherbrooke, QC J1K 2R1, Canada
sabri.boutemedjet@usherbrooke.ca

**Djemel Ziou**
DI, Universite de Sherbrooke
2500 boulevard de l'Université
Sherbrooke, QC J1K 2R1, Canada
djemel.ziou@usherbrooke.ca

**Nizar Bouguila**
CIISE, Concordia University
1515 Ste-Catherine Street West
Montreal, QC H3G 1T7, Canada
bouguila@ciise.concordia.ca

## Abstract

Content-based image suggestion (CBIS) targets the recommendation of products based on user preferences on the visual content of images. In this paper, we motivate both feature selection and model order identification as two key issues for a successful CBIS. We propose a generative model in which the visual features and users are clustered into separate classes. We identify the number of both user and image classes with the simultaneous selection of relevant visual features using the message length approach. The goal is to ensure an accurate prediction of ratings for multidimensional non-Gaussian and continuous image descriptors. Experiments on a collected data have demonstrated the merits of our approach.

## 1 Introduction

Products in today's e-market are described using both visual and textual information. From consumer psychology, the visual information has been recognized as an important factor that influences the consumer's decision making and has an important power of persuasion [4]. Furthermore, it is well recognized that the consumer choice is also influenced by the external environment or context such as the time and location [4]. For example, a consumer could express an information need during a travel that is different from the situation when she or he is working or even at home. *"Content-Based Image Suggestion"* (CBIS) [4] motivates the modeling of user preferences with respect to visual information under the influence of the context. Therefore, CBIS aims at the suggestion of products whose relevance is inferred from the history of users in different contexts on images of the previously consumed products. The domains considered by CBIS are a set of users $\mathcal{U} = \{1, 2, \ldots, N_u\}$, a set of visual documents $\mathcal{V} = \{\vec{v}_1, \vec{v}_2, \ldots, \vec{v}_{N_v}\}$, and a set of possible contexts $\mathcal{E} = \{1, 2, \ldots, N_e\}$. Each $\vec{v}_k$ is an arbitrary descriptor (visual, textual, or categorical) used to represent images or products. In this work, we consider an image as a $D$-dimensional vector $\vec{v} = (v_1, v_2, \ldots, v_D)$. The visual features may be local such as interest points or global such as color, texture, or shape. The relevance is expressed explicitly on an ordered voting (or rating) scale defined as $\mathcal{R} = \{r_1, r_2, \ldots, r_{N_r}\}$. For example, the five star scale (i.e. $N_r = 5$) used by Amazon allows consumers to give different degrees of appreciation. The history of each user $u \in \mathcal{U}$, is defined as $\mathcal{D}^u = \{< u, e^{(j)}, \vec{v}^{(j)}, r^{(j)} > |e^{(j)} \in \mathcal{E}, \vec{v}^{(j)} \in \mathcal{V}, r^{(j)} \in \mathcal{R}, j = 1, \ldots, |D^u|\}$.

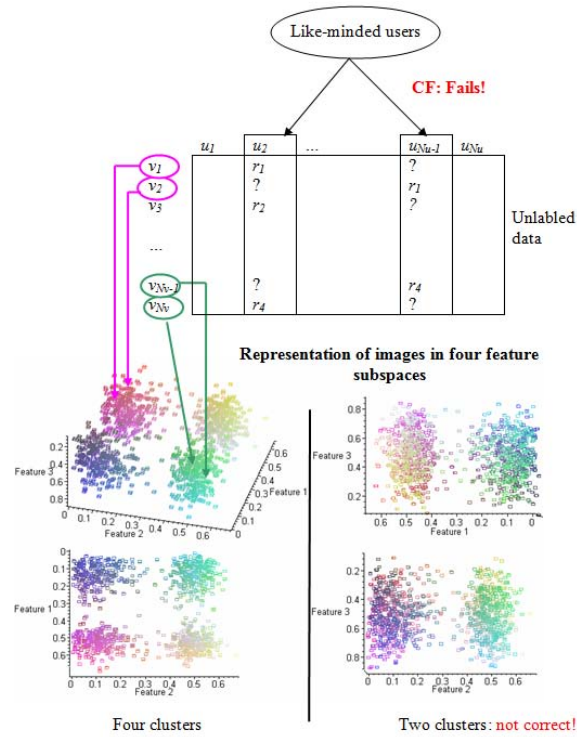

Figure 1: The VCC-FMM identifies like-mindedness from similar appreciations on similar images represented in 3-dimensional space. Notice the inter-relation between the number of image clusters and the considered feature subset.

In literature, the modeling of user preferences has been addressed mainly within collaborative filtering (CF) and content-based filtering (CBF) communities. On the one hand, CBF approaches [12] build a separate model of "liked" and "disliked" discrete data (word features) from each $\mathcal{D}^u$ taken individually. On the other hand, CF approaches predict the relevance of a given product for a given user based on the preferences provided by a set of "like-minded" (similar tastes) users. The data set used by CF is the user-product matrix $(\cup_{u=1}^{N_u} \mathcal{D}^u)$ which is discrete since each product is represented by a categorical index. The Aspect model [7] and the flexible mixture model (FMM) [15] are examples of some model-based CF approaches. Recently, the authors in [4] have proposed a statistical model for CBIS which uses both visual and contextual information in modeling user preferences with respect to multidimensional non Gaussian and continuous data. Users with similar preferences are considered in [4] as those who appreciated with similar degrees similar images. Therefore, instead of considering products as categorical variables (CF), visual documents are represented by a richer visual information in the form of a vector of visual features (texture, shape, and interest points). The similarity between images and between user preferences is modeled in [4] through a single graphical model which clusters users and images separately into homogeneous groups in a similar way to the flexible mixture model (FMM) [15]. In addition, since image data are generally non-Gaussian [1], class-conditional distributions of visual features are assumed Dirichlet densities. By this way, the like-mindedness in user preferences is captured at the level of visual features.

Statistical models for CBIS are useful tools in modeling for many reasons. First, once the model is learned from training data (union of user histories), it can be used to "suggest" unknown (possibly unrated) images efficiently i.e. few effort is required at the prediction phase. Second, the model can be updated from new data (images or ratings) in an online fashion in order to handle the changes in either image clusters and/or user preferences. Third, model selection approaches can be employed to identify *"without supervision"* both numbers of user preferences and image clusters (i.e. model order) from the statistical properties of the data. It should be stressed that the unsupervised selection of the model order was not addressed in CF/CBF literature. Indeed, the model order in many well-

founded statistical models such as the Aspect model [7] or FMM [15] was set *"empirically"* as a compromise between the model's complexity and the accuracy of prediction, but not from the data.

From an "image collection modeling" point of view, the work in [4] has focused on modeling user preferences with respect to non-Gaussian image data. However, since CBIS employs generally high-dimensional image descriptors, then the problem of modeling accurately image collections needs to be addressed in order to overcome the curse of dimensionality and provide accurate suggestions. Indeed, the presence of many irrelevant features degrades substantially the performance of the modeling and prediction [6] in addition to the increase of the computational complexity. To achieve a better modeling, we consider feature selection and extraction as another *"key issue"* for CBIS. In literature [6], the process of feature selection in mixture models have not received as much attention as in supervised learning. The main reason is the absence of class labels that may guide the selection process [6]. In this paper, we address the issue of feature selection in CBIS through a new generative model which we call Visual Content Context-aware Flexible Mixture Model (VCC-FMM). Due to the problem of the inter-relation between feature subsets and the model order i.e. different feature subsets correspond to different natural groupings of images, we propose to learn the VCC-FMM from unlabeled data using the Minimum Message Length (MML) approach [16]. The next Section details the VCC-FMM model with an integrated feature selection. After that, we discuss the identification of the model order using the MML approach in Section 3. Experimental results are presented in Section 4. Finally, we conclude this paper by a summary of the work.

## 2   The Visual Content Context Flexible Mixture Model

The data set $\mathcal{D}$ used to learn a CBIS system is the union of all user histories i.e. $\mathcal{D} = \cup_{u \in \mathcal{U}} \mathcal{D}^u$. From this data set we model both like-mindedness shared by user groups as well as the visual and semantic similarity between images [4]. For that end, we introduce two latent variables $z$ and $c$ to label each observation $< u, e, \vec{v}, r >$ with information about user classes and image classes, respectively. In order to make predictions on unseen images, we need to model the joint event $p(\vec{v}, r, u, e) = \sum_{z,c} p(\vec{v}, r, u, e, z, c)$. Then, the rating $r$ for a given user $u$, context $e$ and a visual document $\vec{v}$ can be predicted on the basis of probabilities $p(r|u, e, v)$ that can be derived by conditioning the generative model $p(u, e, v, r)$. We notice that the full factorization of $p(\vec{v}, r, u, e, z, c)$ using the chain rule leads to quantities with a huge number of parameters which are difficult to interpret in terms of the data [4]. To overcome this problem, we make use of some conditional independence assumptions that constitute our statistical approximation of the joint event $p(\vec{v}, r, u, e)$. These assumptions are illustrated by the graphical representation of the model in figure 2. Let $K$ and $M$ be the number of user classes and images classes respectively, an initial model for CBIS can be derived as [4]:

$$p(\vec{v}, r, u, e) = \sum_{z=1}^{K} \sum_{c=1}^{M} p(z)p(c)p(u|z)p(e|z)p(\vec{v}|c)p(r|z, c) \tag{1}$$

The quantities $p(z)$ and $p(c)$ denote the a priori weights of user and image classes. $p(u|z)$ and $p(e|z)$ denote the likelihood of a user and context to belong respectively to the user's class $z$. $p(r|z, c)$ is the probability to sample a rating for a given user class and image class. All these quantities are modeled from discrete data. On the other hand, image descriptors are high-dimensional, continuous and generally non Gaussian data [1]. Thus, the distribution of class-conditional densities $p(\vec{v}|c)$ should be modeled carefully in order to capture efficiently the added-value of the visual information. In this work, we assume that $p(\vec{v}|c)$ is a Generalized Dirichlet distribution (GDD) which is more appropriate than other distributions such as the Gaussian or Dirichlet distributions in modeling image collections [1]. This distribution has a more general covariance structure and provides multiple shapes. The distribution of the $c$-th component $\vec{\Theta}_c^*$ is given by equation (2). The $^*$ superscript is used to denote the unknown *true* GDD distribution.

$$p(\vec{v}|\vec{\Theta}_c^*) = \prod_{l=1}^{D} \frac{\Gamma(\alpha_{cl}^* + \beta_{cl}^*)}{\Gamma(\alpha_{cl}^*)\Gamma(\beta_{cl}^*)} v_l^{\alpha_{cl}^* - 1} (1 - \sum_{k=1}^{l} v_k)^{\gamma_{cl}^*} \tag{2}$$

where $\sum_{l=l}^{D} v_l < 1$ and $0 < v_l < 1$ for $l = 1, \ldots, D$. $\gamma_{cl}^* = \beta_{cl}^* - \alpha_{cl+1}^* - \beta_{cl+1}^*$ for $l = 1, \ldots, D-1$ and $\gamma_D^* = \beta_D^* - 1$. In equation (2) we have set $\vec{\Theta}_c^* = (\alpha_{c1}^*, \beta_{c1}^*, \ldots, \alpha_{cD}^*, \beta_{cD}^*)$. From the mathematical properties of the GDD, we can transform using a geometric transformation the data point

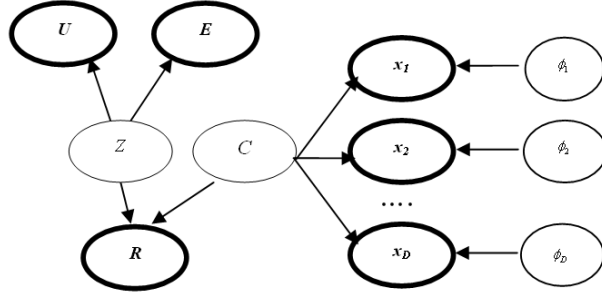

Figure 2: Graphical representation of VCC-FMM.

$\vec{v}$ into another data point $\vec{x} = (x_1, \ldots, x_D)$ with independent features without loss of information [1]. In addition, each $x_l$ of $\vec{x}$ generated by the $c$-th component, follows a Beta distribution $p_b(.|\theta_{cl}^*)$ with parameters $\theta_{cl}^* = (\alpha_{cl}^*, \beta_{cl}^*)$ which leads to the fact $p(\vec{x}|\vec{\Theta}_c^*) = \prod_{l=1}^{D} p_b(x_l|\theta_{cl}^*)$. The independence between $x_l$ makes the estimation of a GDD very efficient i.e. $D$ estimations of univariate Beta distributions without loss of accuracy. However, even with independent features, the unsupervised identification of image clusters based on high-dimensional descriptors remains a hard problem due to the omnipresence of noisy, redundant and uninformative features [6] that degrade the accuracy of the modeling and prediction. We consider feature selection and extraction as a "key" methodology in order to remove that kind of features in our modeling. Since $x_l$ are independent, then we can extract *"relevant"* features in the representation space $\mathcal{X}$. However, we need some definition of feature's relevance. From figure 1, four well-separated image clusters can be identified from only two relevant features 1 and 2 which are multimodal and influenced by class labels. On the other hand, feature 3 is unimodal (i.e. irrelevant) and can be approximated by a single Beta distribution $p_b(.|\xi_l)$ common to all components. This definition of feature's relevance has been motivated in unsupervised learning [2][9]. Let $\vec{\phi} = (\phi_1, \ldots, \phi_D)$ be a set of missing binary variables denoting the relevance of all features. $\phi_l$ is set to 1 when the $l$-th feature is relevant and 0 otherwise. The "true" Beta distribution $\theta_{cl}^*$ can be approximated as [2][9]:

$$p(x_l|\theta_{cl}^*, \phi_l) \simeq \left( p_b(x_l|\theta_{cl}) \right)^{\phi_l} \left( p_b(x_l|\xi_l) \right)^{1-\phi_l} \qquad (3)$$

By considering each $\phi_l$ as Bernoulli variable with parameters $p(\phi_l = 1) = \epsilon_{l_1}$ and $p(\phi_l = 0) = \epsilon_{l_2}$ ($\epsilon_{l_1} + \epsilon_{l_2} = 1$) then, the distribution $p(x_l|\theta_{cl}^*)$ can be obtained after marginalizing over $\phi_l$ [9] as: $p(x_l|\theta_{cl}^*) \simeq \epsilon_{l_1} p_b(x_l|\theta_{cl}) + \epsilon_{l_2} p_b(x_l|\xi_l)$. The VCC-FMM model is given by equation (4). We notice that both models [3] [4] are special cases of VCC-FMM.

$$p(\vec{x}, r, u, e) = \sum_{z=1}^{K} \sum_{c=1}^{M} p(z)p(u|z)p(e|z)p(c)p(r|z,c) \prod_{l=1}^{D} [\epsilon_{l_1} p_b(x_l|\theta_{cl}) + \epsilon_{l_2} p_b(x_l|\xi_l)] \qquad (4)$$

## 3 A Unified Objective for Model and Feature Selection using MML

We denote by $\vec{\theta}_\pi^A$ the parameter vector of the multinomial distribution of any discrete variable $A$ conditioned on its parent $\Pi$ of VCC-FMM (see figure 2). We have $A|_{\Pi=\pi} \sim Multi(1; \vec{\theta}_\pi^A)$ where $\theta_{\pi a}^A = p(A = a|\Pi = \pi)$ and $\sum_a \theta_{\pi a}^A = 1$. Also, we employ the superscripts $\theta$ and $\xi$ to denote the parameters of the Beta distribution of relevant and irrelevant components, respectively $\left(\text{i.e. } \theta_{cl} = (\alpha_{cl}^\theta, \beta_{cl}^\theta) \text{ and } \xi_l = (\alpha_l^\xi, \beta_l^\xi)\right)$. The set $\Theta$ of all VCC-FMM parameters is defined by $\vec{\theta}_z^U, \vec{\theta}_z^E, \vec{\theta}_{zc}^R, \vec{\theta}^{\phi_l}$, $\vec{\theta}^Z, \vec{\theta}^C$ and $\theta_{cl}, \xi_l$. The log-likelihood of a data set of $N$ independent and identically distributed observations $\mathcal{D} = \{< u^{(i)}, e^{(i)}, \vec{x}^{(i)}, r^{(i)} > | i = 1, \ldots, N, u^{(i)} \in \mathcal{U}, e^{(i)} \in \mathcal{E}, \vec{x}^{(i)} \in \mathcal{X}, r^{(i)} \in \mathcal{R}\}$ is given by:

$$\log p(\mathcal{D}|\Theta) = \sum_{i=1}^{N} \log \sum_{z=1}^{K} \sum_{c=1}^{M} p(z)p(c)p(u^{(i)}|z)p(e^{(i)}|z)p(r^{(i)}|z,c) \prod_{l=1}^{D} [\epsilon_{l_1} p_b(x_l^{(i)}|\theta_{cl}) + \epsilon_{l_2} p_b(x_l^{(i)}|\xi_l)]$$

$$(5)$$

The maximum likelihood (ML) approach which optimizes equation (5) w.r.t $\Theta$ is not appropriate for learning VCC-FMM since both $K$ and $M$ are unknown. In addition, the likelihood increases monotonically with the number of components and favors lower dimensions [5]. To overcome these problems, we define a message length objective [16] for both the estimation of $\Theta$ and identification of $K$ and $M$ using MML [9][2]. This objective incorporates in addition to the log-likelihood, a penalty term which encodes the data to penalize complex models as:

$$MML(K,M) = -\log p(\Theta) + \frac{1}{2}\log|I(\Theta)| + \frac{s}{2}(1 + \log\frac{1}{12}) - \log p(\mathcal{D}|\Theta) \qquad (6)$$

In equation (6), $|I(\Theta)|$, $p(\Theta)$, and $s$ denote the Fisher information, prior distribution and the total number of parameters, respectively. The Fisher information of a parameter is the expectation of the second derivatives with respect to the parameter of the minus log-likelihood. It is common sense to assume an independence among the different groups of parameters which factorizes both $|I(\Theta)|$ and $p(\Theta)$ over the Fisher and prior distribution of different groups of parameters, respectively. We approximate the Fisher information of the VCC-FMM from the complete likelihood which assumes the knowledge about the values of hidden variables for each observation $< u^{(i)}, e^{(i)}, \vec{x}^{(i)}, r^{(i)} > \in \mathcal{D}$. The Fisher information of $\theta_{cl}$ and $\xi_l$ can be computed by following a similar methodology of [1]. Also, we use the result found in [8] in computing the Fisher information of $\vec{\theta}_\pi^A$ of a discrete variable $A$ with $N_A$ different values in a data set of $N$ observations. $|I(\vec{\theta}_\pi^A)|$ is given by $|I(\vec{\theta}_\pi^A)| = \left(Np(\Pi = \pi)\right)^{N_A - 1}/\prod_{a=1}^{N_A}\theta_{\pi a}^A$ [8], where $p(\Pi = \pi)$ is the marginal probability of the parent $\Pi$. The graphical representation of of VCC-FMM does not involve variable ancestors (parents of parents). Therefore, the marginal probabilities $p(\Pi = \pi)$ are simply the parameters of the multinomial distribution of the parent variable. For example, $|I(\vec{\theta}_{zc}^R)|$ is computed as: $|I(\vec{\theta}_{zc}^R)| = \left[N^{N_r - 1}(\theta_c^C\theta_z^Z)^{N_r - 1}\right]/\prod_{r=1}^{N_r}\theta_{zcr}^R$. In case of complete ignorance, it is common to employ the Jeffrey's prior for different groups of parameters. Replacing $p(\Theta)$ and $I(\Theta)$ in (6), and after discarding the first order terms, the MML objective is given by:

$$MML(K,M) = \frac{N_p}{2}\log N + M\sum_{l=1}^{D}\log\epsilon_{l_1} + \sum_{l=1}^{D}\log\epsilon_{l_2} + \frac{1}{2}N_p^Z\sum_{z=1}^{K}\log\theta_z^Z$$
$$+ \frac{1}{2}(N_r - 1)\sum_{c=1}^{M}\log\theta_c^C - \log p(\mathcal{D}|\Theta) \qquad (7)$$

with $N_p = 2D(M+1) + K(N_u + N_e - 2) + MK(N_r - 1)$ and $N_p^Z = N_r + N_u + N_e - 3$. For fixed values of $K$, $M$ and $D$, the minimization of MML objective with respect to $\Theta$ is equivalent to a maximum a posteriori (MAP) estimate with the following improper Dirichlet priors [9]:

$$p(\vec{\theta}^C) \propto \prod_{c=1}^{M}(\theta_c^C)^{-\frac{N_r - 1}{2}}, \quad p(\vec{\theta}^Z) \propto \prod_{z=1}^{K}(\theta_z^Z)^{-\frac{N_p^Z}{2}}, \quad p(\epsilon_1, \ldots, \epsilon_D) \propto \prod_{l=1}^{D}\epsilon_{l_1}^{-M}\epsilon_{l_2}^{-1} \qquad (8)$$

## 3.1 Estimation of parameters

We optimize the MML of the data set using the Expectation-Maximization (EM) algorithm in order to estimate the parameters. In the E-step, the joint posterior probabilities of the latent variables given the observations are computed as $Q_{zci} = p(z, c|u^{(i)}, e^{(i)}, \vec{x}^{(i)}, r^{(i)}, \hat{\Theta})$:

$$Q_{zci} = \frac{\hat{\theta}_z^Z\hat{\theta}_c^C\hat{\theta}_{zu^{(i)}}^U\hat{\theta}_{ze^{(i)}}^E\hat{\theta}_{zcr^{(i)}}^R\prod_l(\epsilon_{l_1}p(x_l^{(i)}|\hat{\theta}_{cl}) + \epsilon_{l_2}p(x_l^{(i)}|\hat{\xi}_l))}{\sum_{z,c}\hat{\theta}_z^Z\hat{\theta}_c^C\hat{\theta}_{zu^{(i)}}^U\hat{\theta}_{ze^{(i)}}^E\hat{\theta}_{zcr^{(i)}}^R\prod_l(\epsilon_{l_1}p(x_l^{(i)}|\hat{\theta}_{cl}) + \epsilon_{l_2}p(x_l^{(i)}|\hat{\xi}_l))} \qquad (9)$$

In the M-step, the parameters are updated using the following equations:

$$\hat{\theta}_z^Z = \frac{\max\left(\sum_i\sum_c Q_{zci} - \frac{N_p^Z}{2}, 0\right)}{\sum_z\max\left(\sum_i\sum_c Q_{zci} - \frac{N_p^Z}{2}, 0\right)}, \quad \hat{\theta}_c^C = \frac{\max\left(\sum_i\sum_z Q_{zci} - \frac{N_r - 1}{2}, 0\right)}{\sum_c\max\left(\sum_i\sum_z Q_{zci} - \frac{N_r - 1}{2}, 0\right)} \qquad (10)$$

$$\hat{\theta}_{zu}^U = \frac{\sum_{i:u^{(i)}=u}\sum_c Q_{zci}}{N\hat{\theta}_z^Z}, \quad \hat{\theta}_{ze}^E = \frac{\sum_{i:e^{(i)}=e}\sum_c Q_{zci}}{N\hat{\theta}_z^Z} \quad \hat{\theta}_{zcr}^R = \frac{\sum_{i:r^{(i)}=r}Q_{zci}}{\sum_i Q_{zci}} \qquad (11)$$

$$\frac{1}{\epsilon_{l_1}} = 1 + \frac{\max\left(\sum_{z,c,i}\frac{Q_{zci}\epsilon_{l_2}p_b(x_l^{(i)}|\xi_l)}{\epsilon_{l_1}p_b(x_l^{(i)}|\theta_{cl}) + \epsilon_{l_2}p_b(x_l^{(i)}|\xi_l)} - 1, 0\right)}{\max\left(\sum_{z,c,i}\frac{Q_{zci}\epsilon_{l_1}p_b(X_{il}|\theta_{cl})}{\epsilon_{l_1}p_b(x_l^{(i)}|\theta_{cl}) + \epsilon_{l_2}p_b(x_l^{(i)}|\xi_l)} - M, 0\right)} \qquad (12)$$

The parameters of Beta distributions $\theta_{cl}$ and $\xi_l$ are updated using the Fisher scoring method based on the first and second order derivatives of the MML objective [1].

## 4 Experiments

The benefits of using feature selection and the contextual information are evaluated by considering two variants: V-FMM and V-GD-FMM in addition the original VCC-FMM given by equation (4). V-FMM does not handle the contextual information and assumes $\theta_{ze}^E$ constant for all $e \in \mathcal{E}$. On the other hand, feature selection is not considered for V-GD-FMM by setting $\epsilon_{l_1} = 1$ and pruning the uninformative components $\xi_l$ for $l = 1, \ldots, D$.

### 4.1 Data Set

We have collected ratings from 27 subjects who participated in the experiment (i.e. $N_u = 27$) during a period of three months. The participating subjects are graduate students in faculty of science. Subjects received periodically (twice a day) a list of three images on which they assign relevance degrees expressed on a five star rating scale (i.e. $N_r = 5$). We define the context as a combination of two attributes: location $\mathcal{L} = \{in-campus, out-campus\}$ inferred from the Internet Protocol (IP) address of the subject, and time as $\mathcal{T} = (weekday, weekend)$ i.e $N_e = 4$. A data set $\mathcal{D}$ of 13446 ratings is collected ($N = 13446$). We have used a collection of 4775 (i.e. $N_v = 4775$) images collected from Washington University [10] and collections of free photographs which we categorized manually into 41 categories. For visual content characterization, we have employed both local and global descriptors. For local descriptors, we use the 128-dimensional Scale Invariant Feature Transform (SIFT) [11] to represent image patches. We employ vector quantization to SIFT descriptors and we build a histogram for each image ("bag of visual words"). The size of the visual vocabulary is 500. For global descriptors, we used the color correlogram for image texture representation, and the edge histogram descriptor. Therefore, a visual feature vector is represented in a 540-dimensional space ($D = 540$). We measure the accuracy of the prediction by the Mean Absolute Error (MAE) which is the average of the absolute deviation between the actual and predicted ratings.

### 4.2 First Experiment: Evaluating the influence of model order on the prediction accuracy

This experiment tries to investigate the relationship between the assumed model order defined by $K$ and $M$ on the prediction accuracy of VCC-FMM. It should be noticed that the ground truth number of user classes $K^*$ is not known for our data set $\mathcal{D}$. We run this experiment on a ground truth (artificial) data with known $K$ and $M$. $\mathcal{D}_{GT}$ is sampled from the preferences $P_1$ and $P_2$ of two most dissimilar subjects according to Pearson correlation coefficients [14]. We sample ratings for 100 simulated users from the preferences $P_1$ and $P_2$ only on images of four image classes. For each user, we generate 80 ratings ($\sim$ 20 ratings per context). Therefore, the ground truth model order is $K^* = 2$ and $M^* = 4$. The choice of $M^*$ is purely motivated by convenience of presentation since similar performance was reported for higher values of $M^*$. We learn the VCC-FMM model using one half of $\mathcal{D}_{GT}$ for different choices of training and validation data. The model order defined by $M = 15$ and $K = 15$ is used to initialize EM algorithm.

Figure 3(a) shows that both $K$ and $M$ have been identified correctly on $\mathcal{D}_{GT}$ since the lowest MML was reported for the model order defined by $M = 4$ and $K = 2$. The selection of the best model order is important since it influences the accuracy of the prediction (MAE) as illustrated by Figure 3(b). It should be noticed that the over-estimation of $M$ ($M > M^*$) leads to more errors than the over-estimation of $K$ ($K > K^*$).

### 4.3 Second Experiment: Comparison with state-of-the-art

The aim of this experiment is to measure the contribution of the visual information and the user's context in making accurate predictions comparatively with some existing CF approaches. We make comparisons with the Aspect model [7], Pearson Correlation (PCC)[14], Flexible Mixture Model (FMM) [15], and User Rating Profile (URP) [13]. For accurate estimators, we learn the URP model using Gibs sampling. We retained for the previous algorithms, the model order that ensured the lowest MAE.

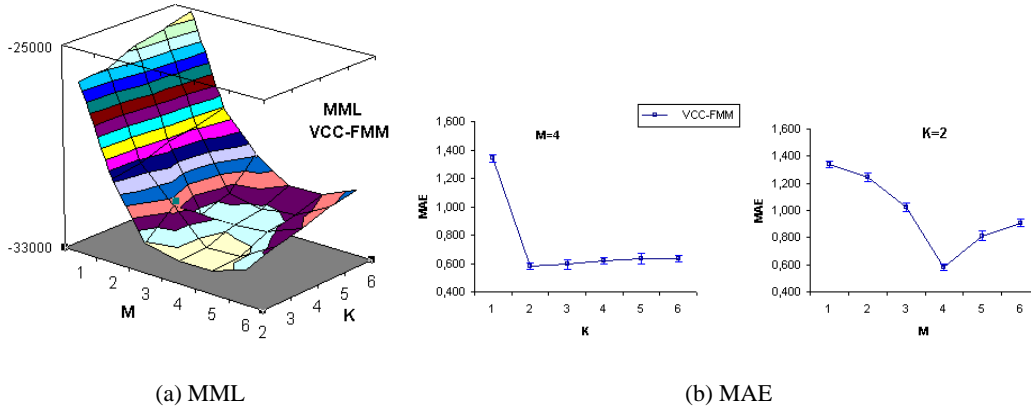

(a) MML                (b) MAE

Figure 3: MML and MAE curves for different model orders on $\mathcal{D}_{GT}$.

Table 1: Averaged MAE over 10 runs of the different algorithms on $\mathcal{D}$

|  | PCC(baseline) | Aspect | FMM | URP | V-FMM | V-GD-FMM | VCC-FMM |
|---|---|---|---|---|---|---|---|
| Avg MAE | 1.327 | 1.201 | 1.145 | 1.116 | 0.890 | 0.754 | 0.646 |
| Deviation | 0.040 | 0.051 | 0.036 | 0.042 | 0.034 | 0.027 | 0.014 |
| Improvement | 0.00% | 9.49% | 13.71% | 15.90% | 32.94% | 43.18% | 55.84% |

The first five columns of table 1 show the added value provided by the visual information comparatively with pure CF techniques. For example, the improvement in the rating's prediction reported by V-FMM is 3.52% and 1.97% comparatively with FMM and URP, respectively. The algorithms (with context information) shown in the last two columns have also improved the accuracy of the prediction comparatively with the others (at least 15.28%). This explains the importance of the contextual information on user preferences. Feature selection is also important since VCC-FMM has reported a better accuracy (14.45%) than V-GD-FMM. Furthermore, it is reported in figure 4(a) that VCC-FMM is less sensitive to data sparsity (number of ratings per user) than pure CF techniques. Finally, the evolution of the average MAE provided VCC-FMM for different proportions of unrated images remains under $< 25\%$ for up to 30% of unrated images as shown in Figure 4(b). We explain the stability of the accuracy of VCC-FMM for data sparsity and new images by the visual information since only cluster representatives need to be rated.

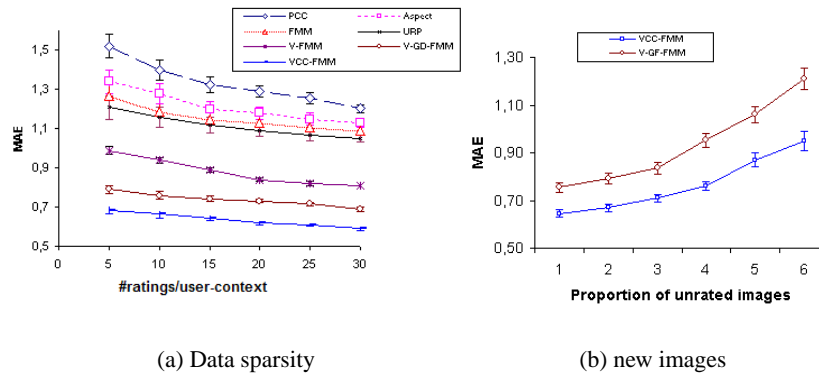

(a) Data sparsity             (b) new images

Figure 4: MAE curves with error bars on the data set $\mathcal{D}$.

# 5   Conclusions

This paper has motivated theoretically and empirically the importance of both feature selection and model order identification from unlabeled data as important issues in content-based image suggestion. Experiments on collected data showed also the importance of the visual information and the user's context in making accurate suggestions.

## Acknowledgements

The completion of this research was made possible thanks to Natural Sciences and Engineering Research Council of Canada (NSERC), Bell Canada's support through its Bell University Laboratories R&D program and a start-up grant from Concordia University.

## References

[1] N. Bouguila and D. Ziou. High-Dimensional Unsupervised Selection and Estimation of a Finite Generalized Dirichlet Mixture Model Based on Minimum Message Length. *IEEE Transactions on Pattern Analysis and Machine Intelligence*, 29(10):1716–1731, 2007.

[2] S. Boutemedjet, N. Bouguila, and D. Ziou. Unsupervised Feature and Model Selection for Generalized Dirichlet Mixture Models. In *Proc. of International Conference on Image Analysis and Recognition (ICIAR)*, pages 330–341. LNCS 4633, 2007.

[3] S. Boutemedjet and D. Ziou. Content-based Collaborative Filtering Model for Scalable Visual Document Recommendation. In *Proc. of IJCAI-2007 Workshop on Multimodal Information Retrieval*, pages 11–18, 2007.

[4] S. Boutemedjet and D. Ziou. A Graphical Model for Context-Aware Visual Content Recommendation. *IEEE Transactions on Multimedia*, 10(1):52–62, 2008.

[5] J. G. Dy and C. E. Brodley. Feature Selection for Unsupervised Learning. *Journal of Machine Learning Research*, 5:845–889, 2004.

[6] I. Guyon and A. Elisseeff. An Introduction to Variable and Feature Selection. *Journal of Machine Learning Research*, 3:1157–1182, 2003.

[7] T. Hofmann. Latent Semantic Models for Collaborative Filtering. *ACM Transactions on Information Systems*, 22(1):89–115, 2004.

[8] P. Kontkanen, P. Myllymki, T. Silander, H. Tirri, and P. Grnwald. On Predictive Distributions and Bayesian Networks. *Statistics and Computing*, 10(1):39–54, 2000.

[9] M. H. C. Law, M.A.T. Figueiredo, and A. K. Jain. Simultaneous Feature Selection and Clustering Using Mixture Models. *IEEE Transactions on Pattern Analysis and Machine Intelligence*, 26(9), 2004.

[10] J. Li and J. Z. Wang. Automatic Linguistic Indexing of Pictures by a Statistical Modeling Approach. *IEEE Transactions on Pattern Analysis and Machine Intelligence*, 25(9):49–68, 2003.

[11] D.G. Lowe. Distinctive Image Features From Scale-Invariant Keypoints. *International Journal of Computer Vision*, 60(2):91–110, 2004.

[12] J. Muramastsu M. Pazzani and D. Billsus. Syskill and Webert:Identifying Interesting Web Sites. In *In Proc. of the 13th National Conference on Artificial Intelligence (AAAI)*, 1996.

[13] B. Marlin. Modeling User Rating Profiles For Collaborative Filtering. In *Proc. of Advances in Neural Information Processing Systems 16 (NIPS)*, 2003.

[14] P. Resnick, N. Iacovou, M. Suchak, P. Bergstrom, and J. Riedl. Grouplens: An Open Architecture for Collaborative Filtering of Netnews. In *Proc. of ACM Conference on Computer Supported Cooperative Work*, 1994.

[15] L. Si and R. Jin. Flexible Mixture Model for Collaborative Filtering. In *Proc. of 20th International Conference on Machine Learning (ICML)*, pages 704–711, 2003.

[16] C. Wallace. *Statistical and Inductive Inference by Minimum Message Length*. Information Science and Statistics. Springer, 2005.
